# A General Boosting Method and its Application to Learning Ranking Functions for Web Search

**Zhaohui Zheng**[†]  **Hongyuan Zha**[⋆]  **Tong Zhang**[†]  **Olivier Chapelle**[†]  **Keke Chen**[†]  **Gordon Sun**[†]

[†]Yahoo! Inc.
701 First Avene
Sunnyvale, CA 94089
`{zhaohui,tzhang,chap,kchen,gzsun}@yahoo-inc.com`

[⋆]College of Computing
Georgia Institute of Technology
Atlanta, GA 30032
`zha@cc.gatech.edu`

## Abstract

We present a general boosting method extending functional gradient boosting to optimize complex loss functions that are encountered in many machine learning problems. Our approach is based on optimization of quadratic upper bounds of the loss functions which allows us to present a rigorous convergence analysis of the algorithm. More importantly, this general framework enables us to use a standard regression base learner such as single regression tree for fitting any loss function. We illustrate an application of the proposed method in learning ranking functions for Web search by combining both preference data and labeled data for training. We present experimental results for Web search using data from a commercial search engine that show significant improvements of our proposed methods over some existing methods.

## 1  Introduction

There has been much interest in developing machine learning methods involving complex loss functions beyond those used in regression and classification problems [13]. Many methods have been proposed dealing with a wide range of problems including ranking problems, learning conditional random fields and other structured learning problems [1, 3, 4, 5, 6, 7, 11, 13]. In this paper we propose a boosting framework that can handle a wide variety of complex loss functions. The proposed method uses a regression black box to optimize a general loss function based on quadratic upper bounds, and it also allows us to present a rigorous convergence analysis of the method. Our approach extends the gradient boosting approach proposed in [8] but can handle substantially more complex loss functions arising from a variety of machine learning problems.

As an interesting and important application of the general boosting framework we apply it to the problem of learning ranking functions for Web search. Specifically, we want to rank a set of documents according to their relevance to a given query. We adopt the following framework: we extract a set of features $x$ for each query-document pair, and learn a function $h(x)$ so that we can rank the documents using the values $h(x)$, say $x$ with larger $h(x)$ values are ranked higher. We call such a function $h(x)$ a ranking function. In Web search, we can identify two types of training data for learning a ranking function: 1) preference data indicating a document is more relevant than another with respect to a query [11, 12]; and 2) labeled data where documents are assigned ordinal labels representing degree of relevancy. In general, we will have both preference data and labeled data for

training a ranking function for Web search, leading to a complex loss function that can be handled by our proposed general boosting method which we now describe.

## 2 A General Boosting Method

We consider the following general optimization problem:

$$\hat{h} = \arg\min_{h \in \mathcal{H}} \mathcal{R}(h), \tag{1}$$

where $h$ denotes a prediction function which we are interested in learning from the data, $\mathcal{H}$ is a pre-chosen function class, and $\mathcal{R}(h)$ is a risk functional with respect to $h$. We consider the following form of the risk functional $\mathcal{R}$:

$$\mathcal{R}(h) = \frac{1}{n} \sum_{i=1}^{n} \phi_i(h(x_{i,1}), \cdots, h(x_{i,m_i}), y_i), \tag{2}$$

where $\phi_i(h_1, \ldots, h_{m_i}, y)$ is a loss function with respect to the first $m_i$ arguments $h_1, \ldots, h_{m_i}$.

For example, each function $\phi_i$ can be a single variable function ($m_i = 1$) such as in regression: $\phi_i(h, y) = (h - y)^2$; or a two-variable function ($m_i = 2$), such as those in ranking based on pairwise comparisons: $\phi_i(h_1, h_2, y) = \max(0, 1 - y(h_1 - h_2))^2$, where $y \in \{\pm 1\}$ indicates whether $h_1$ is preferred to $h_2$ or not; or it can be a multi-variable function as used in some structured prediction problems: $\phi_i(h_1, \ldots, h_{m_i}, y) = \sup_z \delta(y, z) + \psi(h, z) - \psi(h, y)$, where $\delta$ is a loss function [13].

Assume we do not have a general solver for the optimization problem (1), but we have a learning algorithm $\mathcal{A}$ which we refer to as regression weak learner. Given any set of data points $X = [x_1, \ldots, x_k]$, with corresponding target values $R = [r_1, \ldots, r_k]$, weights $W = [w_1, \ldots, w_k]$, and tolerance $\epsilon > 0$, the regression weak learner $\mathcal{A}$ produces a function $\hat{g} = \mathcal{A}(W, X, R, \epsilon) \in \mathcal{C}$ such that

$$\sum_{j=1}^{k} w_j(\hat{g}(x_j) - r_j)^2 \leq \min_{g \in \mathcal{C}} \sum_{j=1}^{k} w_j(g(x_j) - r_j)^2 + \epsilon. \tag{3}$$

Our goal is to use this weak learner $\mathcal{A}$ to solve the original optimization problem (1). Here $\mathcal{H} = \text{span}(\mathcal{C})$, i.e., $h \in \mathcal{H}$ can be expressed as $h(x) = \sum_j a_j h_j(x)$ with $h_j \in \mathcal{C}$.

Friedman [8] proposed a solution when the loss function in (2) can be expressed as

$$\mathcal{R}(h) = \sum_{i=1}^{n} \phi_i(h(x_i)), \tag{4}$$

which he named as gradient boosting. The idea is to estimate the gradient $\nabla \phi_i(h(x_i))$ using regression at each step with uniform weighting, and update. However, there is no convergence proof.

Following his work, we consider an extension that is more principly motivated, for which a convergence analysis can be obtained. We first rewrite (2) in the more general form:

$$\mathcal{R}(h) = \mathcal{R}(h(x_1), \ldots, h(x_N)), \tag{5}$$

where $N \leq \sum m_i$.[1] Note that $\mathcal{R}$ depends on $h$ only through the function values $h(x_i)$ and from now on we identify the function $h$ with the vector $[h(x_i)]$. Also the function $\mathcal{R}$ is considered to be a function of $N$ variables.

Our main observation is that for twice differentiable risk functional $\mathcal{R}$, at each tentative solution $h_k$, we can expand $\mathcal{R}(h)$ around $h_k$ using Taylor expansion as

$$\mathcal{R}(h_k + g) = \mathcal{R}(h_k) + \nabla \mathcal{R}(h_k)^T g + \frac{1}{2} g^T \nabla^2 \mathcal{R}(h') g,$$

where $h'$ lies between $h_k$ and $h_k + g$. The right hand side is almost quadratic, and we can then replace it by a quadratic upper-bound

$$\mathcal{R}(h_k + g) \leq \mathcal{R}_k(g) = \mathcal{R}(h_k) + \nabla \mathcal{R}(h_k)^T g + \frac{1}{2} g^T W g, \tag{6}$$

where $W$ is a diagonal matrix upper bounding the Hessian between $h_k$ and $h_k + g$. If we define $r_j = -[\nabla \mathcal{R}(h_k)]_j / w_j$, then $\forall g \in \mathcal{C}, \sum_j w_j (g(x_j) - r_j)^2$ is equal to the above quadratic form (up to a constant). So $g$ can be found by calling the regression weak learner $\mathcal{A}$. Since at each step we try to minimize an upper bound $\mathcal{R}_k$ of $\mathcal{R}$, if we let the minimum be $g_k$, it is clear that $\mathcal{R}(h_k + g_k) \leq \mathcal{R}_k(g_k) \leq \mathcal{R}(h_k)$. This means that by optimizing with respect to the problem $\mathcal{R}_k$ that can be handled by $\mathcal{A}$, we also make progress with respect to optimizing $\mathcal{R}$. The algorithm based on this idea is listed in Algorithm 1 for the loss function in (5).

Convergence analysis of this algorithm can be established using the idea summarized above; see details in appendix. However, in partice, instead of the quadratic upper bound (which has a theoretical garantee easier to derive), one may also consider minimizing an approximation to the Taylor expansion, which would be closer to a Newton type method.

---

**Algorithm 1** Greedy Algorithm with Quadratic Approximation

---

**Input**: $X = [x_\ell]_{\ell=1,\ldots,N}$
let $h_0 = 0$
**for** $k = 0, 1, 2, \ldots$
    let $W = [w_\ell]_{\ell=1,\ldots,N}$, with either
      $w_\ell = \partial^2 \mathcal{R} / \partial h_k(x_\ell)^2$ or             % Newton-type method with diagonal Hessian
      $W$ global diagonal upper bound on the Hessian     % Upper-bound minimization
    let $R = [r_\ell]_{\ell=1,\ldots,N}$, where $r_\ell = w_\ell^{-1} \partial \mathcal{R} / \partial h_k(x_\ell)$
    pick $\epsilon_k \geq 0$
    let $g_k = \mathcal{A}(W, X, R, \epsilon_k)$
    pick step-size $s_k \geq 0$, typically by line search on $\mathcal{R}$
    let $h_{k+1} = h_k + s_k g_k$
**end**

---

The main conceptual difference between our view and that of Friedman is that he views regression as a "reasonable" approximation to the first order gradient $\nabla \mathcal{R}$, while our work views it as a natural consequence of second order approximation of the objective function (in which the quadratic term serve as an upper bound of the Hessian either locally or globally). This leads to algorithmic difference. In our approach, a good choice of the second order upper bound (leading to tighter bound) may require *non-uniform* weights $W$. This is inline with earlier boosting work in which sample-reweighting was a central idea. In our framework, the reweighting naturally occurs when we choose a tight second order approximation. Different reweighting can affect the rate of convergence in our analysis. The other main difference with Friedman is that he only considered objective functions of the form (4); we propose a natural extension to the ones of the form (5).

## 3 Learning Ranking Functions

We now apply Algorithm 1 to the problem of learning ranking functions. We use preference data as well as labeled data for training the ranking function. For preference data, we use $x \succ y$ to mean that $x$ is preferred over $y$ or $x$ should be ranked higher than $y$, where $x$ and $y$ are the feature vectors for corresponding items to be ranked. We denote the set of available preferences as $\mathcal{S} = \{x_i \succ y_i, \; i = 1, \ldots, N\}$. In addition to the preference data, there are also labeled data, $\mathcal{L} = \{(z_i, l_i), \; i = 1, \ldots, n\}$, where $z_i$ is the feature of an item and $l_i$ is the corresponding numerically coded label.[2] We formulate the ranking problem as computing a ranking function $h \in \mathcal{H}$, such that $h$ satisfies as much as possible the set of preferences, i.e., $h(x_i) \geq h(y_i)$, if $x_i \succ y_i, i = 1, \ldots, N$, while at the same time $h(z_i)$ matches the label $l_i$ in a sense to be detailed below.

THE OBJECTIVE FUNCTION. We use the following objective function to measure the empirical risk of a ranking function $h$,

$$\mathcal{R}(h) = \frac{w}{2} \sum_{i=1}^{N} (\max\{0, h(y_i) - h(x_i) + \tau\})^2 + \frac{1-w}{2} \sum_{i=1}^{n} (l_i - h(z_i))^2.$$

The objective function consists of two parts: 1) for the preference data part, we introduce a margin parameter $\tau$ and would like to enforce that $h(x_i) \geq h(y_i) + \tau$; if not, the difference is quadratically penalized; and 2) for the labeled data part, we simply minimize the squared errors. The parameter $w$ is the relative weight for the preference data and could typically be found by cross-validation.

The optimization problem we seek to solve is $h^* = \text{argmin}_{h \in \mathcal{H}} \mathcal{R}(h)$, where $\mathcal{H}$ is some given function class. Note that $\mathcal{R}$ depends only on the values $h(x_i), h(y_i), h(z_i)$ and we can optimize it using the general boosting framework discussed in section 2.

QUADRATIC APPROXIMATION. To this end consider the quadratic approximation (6) for $\mathcal{R}(h)$. For simplicity let us assume that each feature vector $x_i$, $y_i$ and $z_i$ only appears in $\mathcal{S}$ and $\mathcal{L}$ once, otherwise we need to compute appropriately formed averages. We consider

$$h(x_i), h(y_i), \quad i = 1, \ldots, N, \quad h(z_i), \quad i = 1, \ldots, n$$

as the unknowns, and compute the gradient of $\mathcal{R}(h)$ with respect to those unknowns. The components of the *negative* gradient corresponding to $h(z_i)$ is just $l_i - h(z_i)$. The components of the *negative* gradient corresponding to $h(x_i)$ and $h(y_i)$, respectively, are

$$\max\{0, h(y_i) - h(x_i) + \tau\}, \quad -\max\{0, h(y_i) - h(x_i) + \tau\}.$$

Both of the above equal to zero when $h(x_i) - h(y_i) \geq \tau$. For the second-order term, it can be readily verified that the Hessian of $\mathcal{R}(h)$ is block-diagonal with 2-by-2 blocks corresponding to $h(x_i)$ and $h(y_i)$ and 1-by-1 blocks for $h(z_i)$. In particular, if we evaluate the Hessian at $h$, the 2-by-2 block equals to

$$\begin{bmatrix} 1 & -1 \\ -1 & 1 \end{bmatrix}, \quad \begin{bmatrix} 0 & 0 \\ 0 & 0 \end{bmatrix},$$

for $x_i \succ y_i$ with $h(x_i) - h(y_i) < \tau$ and $h(x_i) - h(y_i) \geq \tau$, respectively. We can upper bound the first matrix by the diagonal matrix $\text{diag}(2,2)$ leading to a quadratic upper bound. We summarize the above derivations in the following algorithm.

---

**Algorithm 2** Boosted Ranking using Successive Quadratic Approximation (QBRank)

---

Start with an initial guess $h_0$, for $m = 1, 2, \ldots,$
  1) we construct a training set for fitting $g_m(x)$ by adding the following for each $\langle x_i, y_i \rangle \in \mathcal{S}$,
  $(x_i, \max\{0, h_{m-1}(y_i) - h_{m-1}(x_i) + \tau\})$, $(y_i, -\max\{0, h_{m-1}(y_i) - h_{m-1}(x_i) + \tau\})$,
  and
    $\{(z_i, l_i - h_{m-1}(z_i)), \quad i = 1, \ldots, n\}.$
  The fitting of $g_m(x)$ is done by using a base regressor with the above training set; We weigh the above preference data by $w$ and the labeled data by $1 - w$ respectively.
  2) forming $h_m = h_{m-1} + \eta s_m g_m(x)$,
  where $s_m$ is found by line search to minimize the objective function. $\eta$ is a shrinkage factor.

---

The shrinkage factor $\eta$ by default is 1, but Friedman [8] reported better results (coming from better regularization) by taking $\eta < 1$. In general, we choose $\eta$ and $w$ by cross-validation. $\tau$ could be the degree of preference if that information is available, e.g., the absolute grade difference between each prefernce if it is converted from labeled data. Otherwise, we simply set it to be 1.0. When there is no preference data and the weak regression learner produces a regression tree, QBrank is identical to Gradient Boosting Trees (GBT) as proposed in [8].

REMARK. An $x_i$ can appear multiple times in Step 1), in this case we use the average gradient values as the target value for each distinct $x_i$.

## 4 Experiment Results

We carried out several experiments illustrating the properties and effectiveness of QBrank using combined preference data and labeled data in the context of learning ranking functions for Web search [3]. We also compared its performance with QBrank using preference data only and several existing algorithms such as Gradient Boosting Trees [8] and RankSVM [11, 12]. RankSVM is a preference learning method which learns pair-wise preferences based on SVM approach.

DATA COLLECTION. We first describe how the data used in the experiments are collected. For each query-document pair we extracted a set of features to form a feature vector. which consists of three parts, $x = [x^Q, x^D, x^{QD}]$, where 1) the query-feature vector $x^Q$ comprises features dependent on the query $q$ only and have constant values across all the documents $d$ in the document set, for example, the number of terms in the query, whether or not the query is a person name, etc.; 2) the document-feature vector $x^D$ comprises features dependent on the document $d$ only and have constant values across all the queries $q$ in the query set, for example, the number of inbound links pointing to the document, the amount of anchor-texts in bytes for the document, and the language identity of the document, etc.; and 3) the query-document feature vector $x^{QD}$ which comprises features dependent on the relation of the query $q$ with respect to the document $d$, for example, the number of times each term in the query $q$ appears in the document $d$, the number of times each term in the query $q$ appears in the anchor-texts of the document $d$, etc.

We sampled a set of queries from the query logs of a commercial search engine and generated a certain number of query-document pairs for each of the queries. A five-level numerical grade $(0, 1, 2, 3, 4)$ is assigned to each query-document pair based on the degree of relevance. In total we have 4,898 queries and 105,243 query-document pairs. We split the data into three subsets as follows: 1) we extract all the queries which have documents with a single label. The set of feature vectors and the corresponding labels form training set $L_1$, which contains around 2000 queries giving rise to 20,000 query-document pairs. (Some single-labeled data are from editorial database, where each query has a few ideal results with the same label. Other are bad ranking cases submitted internally and all the documents for a query are labeled as bad. As we will see those type of single-labeled data are very useful for learning ranking functions); and 2) we then randomly split the remaining data by queries, and construct a training set $L_2$ containing about 1300 queries and 40,000 query-document pairs and a test set $L_3$ with about 1400 queries and 44,000 query-document pairs.

We use $L_2$ or $L_3$ to generate a set of preference data as follows: given a query $q$ and two documents $d_x$ and $d_y$. Let the feature vectors for $(q, d_x)$ and $(q, d_y)$ be $x$ and $y$, respectively. If $d_x$ has a higher grade than $d_y$, we include the preference $x \succ y$ while if $d_y$ has a higher grade than $d_x$, we include the preference $y \succ x$. For each query, we consider all pairs of documents within the search results for that query except those with equal grades. This way, we generate around 500,000 preference pairs in total. We denote the preference data as $P_2$ and $P_3$ corresponding to $L_2$ and $L_3$, respectively.

EVALUATION METRICS. The output of QBrank is a ranking function $h$ which is used to rank the documents $x$ according to $h(x)$. Therefore, document $x$ is ranked higher than $y$ by the ranking function $h$ if $h(x) > h(y)$, and we call this the predicted preference. We propose the following two metrics to evaluate the performance of a ranking function with respect to a given set of preferences which we considered as the true preferences.

1) Precision at $K$%: for two documents $x$ and $y$ (with respect to the same query), it is reasonable to assume that it is easy to compare $x$ and $y$ if $|h(x) - h(y)|$ is large, and $x$ and $y$ should have about the same rank if $h(x)$ is close to $h(y)$. Base on this, we sort all the document pairs $\langle x, y \rangle$ according to $|h(x) - h(y)|$. We call *precision at $K$%*, the fraction of non-contradicting pairs in the top $K$% of the sorted list. Precision at 100% can be considered as an overall performance measure of a ranking function.

2) Discounted Cumulative Gain (DCG): DCG has been widely used to assess relevance in the context of search engines [10]. For a ranked list of $N$ documents ($N$ is set to be 5 in our experiments), we use the following variation of DCG, $\text{DCG}_N = \sum_{i=1}^{N} G_i / \log_2(i+1)$, where $G_i$ represents the weights assigned to the label of the document at position $i$. Higher degree of relevance corresponds to higher value of the weight.

PARAMETERS. There are three parameters in QBrank: $\tau$, $\eta$, and $w$. In our experiments, $\tau$ is the absolute grade difference between each pair $\langle x_i, y_i \rangle$. We set $\eta$ to be 0.05, and $w$ to be 0.5 in our

Table 1: Precision at $K$% for QBrank, GBT, and RankSVM

| %K | QBrank | GBT | RankSVM |
|------|--------|--------|---------|
| 10% | 0.9446 | 0.9328 | 0.8524 |
| 20% | 0.903 | 0.8939 | 0.8152 |
| 30% | 0.8611 | 0.8557 | 0.7839 |
| 40% | 0.8246 | 0.8199 | 0.7578 |
| 50% | 0.7938 | 0.7899 | 0.7357 |
| 60% | 0.7673 | 0.7637 | 0.7151 |
| 70% | 0.7435 | 0.7399 | 0.6957 |
| 80% | 0.7218 | 0.7176 | 0.6779 |
| 90% | 0.7015 | 0.6977 | 0.6615 |
| 100% | 0.6834 | 0.6803 | 0.6465 |

experiments. For a fair comparsion, we used single regression tree with 20 leaf nodes as the base regressor of both GBT and QBrank in our experiments. $\eta$ and number of leaf nodes were tuned for GBT through cross validation. We did not retune them for QBrank.

EXPERIMENTS AND RESULTS. We are interested in the following questions: 1) How does GBT using labeled data $L_2$ compare with QBrank or RankSVM using the preference data extracted from the same labeled data: $P_2$? and 2) Is it useful to include single-labeled data $L_1$ in GBT and QBrank? To this end, we considered the following six experiments for comparison: 1) GBT using $L_1$, 2) GBT using $L_2$, 3) GBT using $L_1 \cup L_2$, 4) RankSVM using $P_2$, 5) QBrank using $P_2$, and 6) QBrank using $P_2 \cup L_1$.

Table 1 presents the precision at $K$% on data $P_3$ for the ranking function learned from GBT with labeled training data $L_2$, and QBrank and RankSVM with the corresponding preference data $P_2$. This shows that QBrank outperforms both GBT and RankSVM with respect to the precision at $K$% metric.

The DCG-5 for RankSVM using $P_2$ is 6.181 while that for the other five methods are shown in Figure 1, from which we can see it is useful to include single-labeled data in GBT training. In case of preference learning, no preference pairs could be extracted from single labeled data. Therefore, existing methods such as RankSVM, RankNet and RankBoost that are formulated for preference data only can not take advantage of such data. The QBrank framework can combine preference data and labeled data in a natural way. From Figure 1, we can see QBrank using combined preference data and labeled data outperforms both QBrank and RankSVM using preference data only, which indicates that singled labeled data are also useful to QBrank training. Another observation is that GBT using labeled data is significantly worse than QBrank using preference data extracted from the same labeled data[3]. The clear convergence trend of QBrank is also demonstrated in Figure 1. Notice that, we excluded all tied data (pairs of documents with the same grades) when converting preference data from the absolute relevance judgments, which can be significant information loss, for example of $x_1 > x_2$, and $x_3 > x_4$. If we know $x_2$ ties with $x_3$, then we can have the whole ranking $x_1 > \{x_2, x_3\} > x_4$. Including tied data could further improve performance of both GBrank and QBrank.

## 5   Conclusions and Future Work

We proposed a general boosting method for optimizing complex loss functions. We also applied the general framework to the problem of learning ranking functions. Experimental results using a commercial search engine data show that our approach leads to significant improvements. In future work, 1) we will add regularization to the preference part in the objective function; 2) we plan to apply our general boosting method to other structured learning problems; and 3) we will also explore other applications where both preference and labeled data are available for training ranking functions.

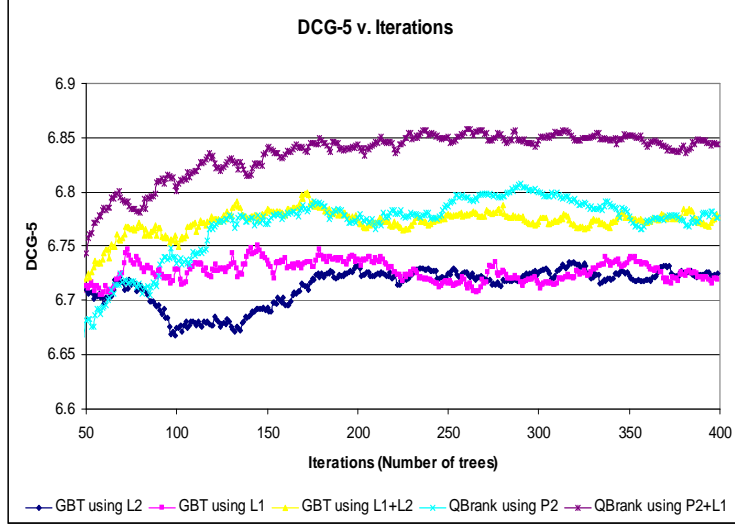

Figure 1: DCG v. Iterations. Notice that DCG for RankSVM using $P_2$ is 6.181.

## Appendix: Convergence results

We introduce a few definitions.

**Definition 1** *$\mathcal{C}$ is scale-invariant if $\forall g \in \mathcal{C}$ and $\alpha \in R$, $\alpha g \in \mathcal{C}$.*

**Definition 2** *$\|g\|_{W,X} = \sqrt{\frac{1}{n} \sum_\ell w_\ell g(x_\ell)^2}$.*

**Definition 3** *Let $h \in \mathrm{span}(\mathcal{C})$, then $\|h\|_{W,X} = \inf\left\{ \sum_j |\alpha_j| : h = \sum_j \alpha_j g_j / \|g_j\|_{W,X}; g_j \in \mathcal{C} \right\}$.*

**Definition 4** *Let $R(h)$ be a function of $h$, an global upper bound $M$ of its Hessian with respect to $[W, X]$ satisfy: $\forall h, \beta$ and $g$: $R(h + \beta g) \leq R(h) + \beta \nabla R(h)^T g + \frac{\beta^2}{2} M \|g\|_{W,X}^2$.*

Although we only consider global upper bounds, it is easy to see that results with respect to local upper bounds can also be established.

**Theorem 1** *Consider Algorithm 1, where $R$ is a convex function of $h$. Let $M$ be an upper bound of the Hessian of $R$. Assume that $\mathcal{C}$ is scale-invariant. Let $\bar{h} \in \mathrm{span}(\mathcal{C})$. Let $\bar{s}_k = s_k \|g_k\|_{W,X}$ be the normalized step-size, $a_j = \sum_{i=0}^j \bar{s}_i$, and $b_j = \sum_{i \geq j} (\bar{s}_i \sqrt{2\epsilon_i} + M \bar{s}_i^2/2)$, then*

$$R(h_{k+1}) \leq R(\bar{h}) + \frac{\|\bar{h}\|_{W,X}}{\|\bar{h}\|_{W,X} + a_k} \max(0, R(0) - R(\bar{h})) + \inf_j \left[ (b_0 - b_{j+1}) \frac{\|\bar{h}\|_{W,X} + a_j}{\|\bar{h}\|_{W,X} + a_k} + (b_{j+1} - b_{k+1}) \right].$$

*If we choose $\bar{s}_k \geq 0$ such that $\sum_k \bar{s}_k = \infty$ and $\sum_k (\bar{s}_k^2 + \bar{s}_k \sqrt{\epsilon_k}) < \infty$, then $\lim_{k \to \infty} R(h_k) = \inf_{\bar{h} \in \mathrm{span}(\mathcal{C})} R(\bar{h})$, and the rate of convergence compared to any target $\bar{h} \in \mathrm{span}(\mathcal{C})$ only depends on $\|\bar{h}\|_{W,X}$, and the sequences $\{a_j\}$ and $\{b_j\}$.*

The proof is a systematic application of the idea outlined earlier and will be detailed in a separate publication. In practice, one often set the step size to be a small constant. In particular, for for some fixed $s > 0$, we can choose $\sqrt{2\epsilon_i} \leq M s^2/2$, and $s_k \|g_k\|_{W,X} = s^2$ when $R(h_k + \bar{s}_k \tilde{g}_k) \leq R(h_k)$ ($\bar{s}_k = 0$ otherwise). Theorem 1 gives the following bound when $k \geq \sqrt{\|\bar{h}\|_{W,X} \max(0, R(0) - R(\bar{h}))/M} s^{-3}$,

$$R(h_{k+1}) \leq R(\bar{h}) + 2s\sqrt{\max(0, R(0) - R(\bar{h}))\|\bar{h}\|_{W,X} M} + M s^4.$$

The convergence results show that in order to have a risk not much worse than any target function $\bar{h} \in \operatorname{span}(\mathcal{C})$, the approximation function $h_k$ does not need to be very complex when the complexity is measured by its 1-norm. It is also important to see that quantities appearing in the generalization analysis do not depend on the number of samples. These results imply that statistically, Algorithm 1 (with small step-size) has an implicit regularization effect that prevents the procedure from overfiting the data. Standard empirical process techniques can then be applied to obtain generalization bounds for Algorithm 1.

## Footnotes

[1]We consider that all $x_i$ are different, but some of the $x_{i,m_i}$ in (2) might have been identical, hence the inequality.

[2]Some may argue that, absolute relevance judgments can also be converted to relative relevance judgments. For example, for a query, suppose we have three documents $d_1, d_2$ and $d_3$ labeled as perfect, good, and bad, respectively. We can obtain the following relative relevance judgments: $d_1$ is preferred over $d_2$, $d_1$ is preferred over $d_3$ and $d_2$ is preferred over $d_3$. However, it is often the case in Web search that for many queries there only exist documents with a single label and for such kind of queries, no preference data can be constructed.

[3]a 1% dcg gain is considered signficant on this data set for commercial search engines.

## References

[1] BALCAN N., BEYGELZIMER A., LANGFORD J., AND SORKIN G. Robust Reductions from Ranking to Classification, manuscript, 2007.

[2] BERTSEKAS D. *Nonlinear programming*. Athena Scientific, second edition, 1999.

[3] BURGES, C., SHAKED, T., RENSHAW, E., LAZIER, A., DEEDS, M., HAMILTON, N., AND HULLENDER, G. Learning to rank using gradient descent. *Proc. of Intl. Conf. on Machine Learning (ICML)* (2005).

[4] DIETTERICH, T. G., ASHENFELTER, A., BULATOV, Y. Training Conditional Random Fields via Gradient Tree Boosting *Proc. of Intl. Conf. on Machine Learning (ICML)* (2004).

[5] CLEMENCON S., LUGOSI G., AND VAYATIS N. Ranking and scoring using empirical risk minimization. *Proc. of COLT* (2005).

[6] COHEN, W. W., SCHAPIRE, R. E., AND SINGER, Y. Learning to order things. Journal of Artificial Intelligence Research, *Neural Computation*, 13, 14431472 (1999).

[7] FREUND, Y., IYER, R., SCHAPIRE, R. E., AND SINGER, Y. An efficient boosting algorithm for combining preferences. *Journal of Machine Learning Research 4* (2003), 933–969.

[8] FRIEDMAN, J. H. Greedy function approximation: A gradient boosting machine. *Annals of Statistics 29*, 5 (2001), 1189–1232.

[9] HERBRICH, R., GRAEPEL, T., AND OBERMAYER, K. Large margin rank boundaries for ordinal regression. 115–132.

[10] JARVELIN, K., AND KEKALAINEN, J. Ir evaluation methods for retrieving highly relevant documents. *Proc. of ACM SIGIR Conference* (2000).

[11] JOACHIMS, T. Optimizing search engines using clickthrough data. *Proc. of ACM SIGKDD Conference* (2002).

[12] JOACHIMS, T., GRANKA, L., PAN, B., AND GAY, G. Accurately interpreting clickthough data as implicit feedback. *Proc. of ACM SIGIR Conference* (2005).

[13] TSOCHANTARIDIS, I., JOACHIMS, T., HOFMANN, T., AND ALTUN, Y. Large margin methods for structured and interdependent output variables. *Journal of Machine Learning Research*, 6:1453–1484, 2005.

